# Bayesian Learning
# via Stochastic Dynamics

**Radford M. Neal**
Department of Computer Science
University of Toronto
Toronto, Ontario, Canada   M5S 1A4

## Abstract

The attempt to find a single "optimal" weight vector in conventional network training can lead to overfitting and poor generalization. Bayesian methods avoid this, without the need for a validation set, by averaging the outputs of many networks with weights sampled from the posterior distribution given the training data. This sample can be obtained by simulating a stochastic dynamical system that has the posterior as its stationary distribution.

## 1   CONVENTIONAL AND BAYESIAN LEARNING

I view neural networks as probabilistic models, and learning as statistical inference. Conventional network learning finds a single "optimal" set of network parameter values, corresponding to maximum likelihood or maximum penalized likelihood inference. Bayesian inference instead integrates the predictions of the network over all possible values of the network parameters, weighting each parameter set by its posterior probability in light of the training data.

### 1.1   NEURAL NETWORKS AS PROBABILISTIC MODELS

Consider a network taking a vector of real-valued inputs, $\mathbf{x}$, and producing a vector of real-valued outputs, $\overline{\mathbf{y}}$, perhaps computed using hidden units. Such a network architecture corresponds to a function, $f$, with $\overline{\mathbf{y}} = f(\mathbf{x}, \mathbf{w})$, where $\mathbf{w}$ is a vector of connection weights. If we assume the observed outputs, $\mathbf{y}$, are equal to $\overline{\mathbf{y}}$ plus Gaussian noise of standard deviation $\sigma$, the network defines the conditional probability

for an observed output vector given an input vector as follows:

$$P(\mathbf{y} \mid \mathbf{x}, \sigma) \quad \propto \quad \exp\left(-|\mathbf{y} - f(\mathbf{x}, \mathbf{w})|^2 / 2\sigma^2\right) \tag{1}$$

The probability of the outputs in a training set $(\mathbf{x}_1, \mathbf{y}_1), \ldots, (\mathbf{x}_n, \mathbf{y}_n)$ given this fixed noise level is therefore

$$P(\mathbf{y}_1, \ldots, \mathbf{y}_n \mid \mathbf{x}_1, \ldots, \mathbf{x}_n, \sigma) \quad \propto \quad \exp\left(-\sum_c |\mathbf{y}_c - f(\mathbf{x}_c, \mathbf{w})|^2 / 2\sigma^2\right) \tag{2}$$

Often $\sigma$ is unknown. A Bayesian approach to handling this is to assign $\sigma$ a vague prior distribution and then integrating it away, giving the following probability for the training set (see (Buntine and Weigend, 1991) or (Neal, 1992) for details):

$$P(\mathbf{y}_1, \ldots, \mathbf{y}_n \mid \mathbf{x}_1, \ldots, \mathbf{x}_n) \quad \propto \quad \left(s_0 + \sum_c |\mathbf{y}_c - f(\mathbf{x}_c, \mathbf{w})|^2\right)^{-\frac{m_0 + nD}{2}} \tag{3}$$

where $s_0$ and $m_0$ are parameters of the prior for $\sigma$.

## 1.2 CONVENTIONAL LEARNING

Conventional backpropagation learning tries to find the weight vector that assigns the highest probability to the training data, or equivalently, that minimizes minus the log probability of the training data. When $\sigma$ is assumed known, we can use (2) to obtain the following objective function to minimize:

$$M(\mathbf{w}) \quad = \quad \sum_c |\mathbf{y}_c - f(\mathbf{x}_c, \mathbf{w})|^2 \big/ 2\sigma^2 \tag{4}$$

When $\sigma$ is unknown, we can instead minimize the following, derived from (3):

$$M(\mathbf{w}) \quad = \quad \tfrac{m_0 + nD}{2} \log\left(s_0 + \sum_c |\mathbf{y}_c - f(\mathbf{x}_c, \mathbf{w})|^2\right) \tag{5}$$

Conventional learning often leads to the network *overfitting* the training data — modeling the noise, rather than the true regularities. This can be alleviated by stopping learning when the the performance of the network on a separate *validation* set begins to worsen, rather than improve. Another way to avoid overfitting is to include a *weight decay* term in the objective function, as follows:

$$M'(\mathbf{w}) \quad = \quad \lambda|\mathbf{w}|^2 + M(\mathbf{w}) \tag{6}$$

Here, the data fit term, $M(\mathbf{w})$, may come from either (4) or (5). We must somehow find an appropriate value for $\lambda$, perhaps, again, using a separate validation set.

## 1.3 BAYESIAN LEARNING AND PREDICTION

Unlike conventional training, Bayesian learning does not look for a single "optimal" set of network weights. Instead, the training data is used to find the *posterior* probability distribution over weight vectors. Predictions for future cases are made by averaging the outputs obtained with all possible weight vectors, with each contributing in proportion to its posterior probability.

To obtain the posterior, we must first define a *prior* distribution for weight vectors. We might, for example, give each weight a Gaussian prior of standard deviation $\omega$:

$$P(\mathbf{w}) \quad \propto \quad \exp\left(-|\mathbf{w}|^2 / 2\omega^2\right) \tag{7}$$

We can then obtain the posterior distribution over weight vectors given the training cases $(\mathbf{x}_1, \mathbf{y}_1), \ldots, (\mathbf{x}_n, \mathbf{y}_n)$ using Bayes' Theorem:

$$P(\mathbf{w} \mid (\mathbf{x}_1, \mathbf{y}_1), \ldots, (\mathbf{x}_n, \mathbf{y}_n)) \quad \propto \quad P(\mathbf{w}) \, P(\mathbf{y}_1, \ldots, \mathbf{y}_n \mid \mathbf{x}_1, \ldots, \mathbf{x}_n, \mathbf{w}) \quad (8)$$

Based on the training data, the best prediction for the output vector in a test case with input vector $\mathbf{x}_*$, assuming squared-error loss, is

$$\widehat{\mathbf{y}}_* \quad = \quad \int f(\mathbf{x}_*, \mathbf{w}) \, P(\mathbf{w} \mid (\mathbf{x}_1, \mathbf{y}_1), \ldots, (\mathbf{x}_n, \mathbf{y}_n)) \, d\mathbf{w} \quad (9)$$

A full predictive distribution for the outputs in the test case can also be obtained, quantifying the uncertainty in the above prediction.

## 2    INTEGRATION BY MONTE CARLO METHODS

Integrals such as that of (9) are difficult to evaluate. Buntine and Weigend (1991) and MacKay (1992) approach this problem by approximating the posterior distribution by a Gaussian. Instead, I evaluate such integrals using Monte Carlo methods.

If we randomly select weight vectors, $\mathbf{w}_0, \ldots, \mathbf{w}_{N-1}$, each distributed according to the posterior, the prediction for a test case can be found by approximating the integral of (9) by the average output of networks with these weights:

$$\widehat{\mathbf{y}}_* \quad \approx \quad \frac{1}{N} \sum_t f(\mathbf{x}_*, \mathbf{w}_t) \quad (10)$$

This formula is valid even if the $\mathbf{w}_t$ are dependent, though a larger sample may then be needed to achieve a given error bound. Such a sample can be obtained by simulating an ergodic Markov chain that has the posterior as its stationary distribution. The early part of the chain, before the stationary distribution has been reached, is discarded. Subsequent vectors are used to estimate the integral.

### 2.1    FORMULATING THE PROBLEM IN TERMS OF ENERGY

Consider the general problem of obtaining a sample of (dependent) vectors, $\mathbf{q}_t$, with probabilities given by $P(\mathbf{q})$. For Bayesian network learning, $\mathbf{q}$ will be the weight vector, or other parameters from which the weights can be obtained, and the distribution of interest will be the posterior.

It will be convenient to express this probability distribution in terms of a *potential energy* function, $E(\mathbf{q})$, chosen so that

$$P(\mathbf{q}) \quad \propto \quad \exp(-E(\mathbf{q})) \quad (11)$$

A *momentum* vector, $\mathbf{p}$, of the same dimensions as $\mathbf{q}$, is also introduced, and defined to have a *kinetic energy* of $\frac{1}{2}|\mathbf{p}|^2$. The sum of the potential and kinetic energies is the *Hamiltonian*:

$$H(\mathbf{q}, \mathbf{p}) \quad = \quad E(\mathbf{q}) \; + \; \tfrac{1}{2}|\mathbf{p}|^2 \quad (12)$$

From the Hamiltonian, we define a joint probability distribution over $\mathbf{q}$ and $\mathbf{p}$ (*phase space*) as follows:

$$P(\mathbf{q}, \mathbf{p}) \quad \propto \quad \exp(-H(\mathbf{q}, \mathbf{p})) \quad (13)$$

The marginal distribution for $\mathbf{q}$ in (13) is that of (11), from which we wish to sample.

We can therefore proceed by sampling from this joint distribution for **q** and **p**, and then just ignoring the values obtained for **p**.

## 2.2  HAMILTONIAN DYNAMICS

Sampling from the distribution (13) can be split into two subproblems — first, to sample *uniformly* from a surface where $H$, and hence the probability, is constant, and second, to visit points of differing $H$ with the correct probabilities. The solutions to these subproblems can then be interleaved to give an overall solution.

The first subproblem can be solved by simulating the *Hamiltonian dynamics* of the system, in which **q** and **p** evolve through a fictitious time, $\tau$, according to the following equations:

$$\frac{d\mathbf{q}}{d\tau} = \frac{\partial H}{\partial \mathbf{p}} = \mathbf{p}, \qquad \frac{d\mathbf{p}}{d\tau} = -\frac{\partial H}{\partial \mathbf{q}} = -\nabla E(\mathbf{q}) \tag{14}$$

This dynamics leaves $H$ constant, and preserves the volumes of regions of phase space. It therefore visits points on a surface of constant $H$ with uniform probability.

When simulating this dynamics, some discrete approximation must be used. The *leapfrog* method exactly maintains the preservation of phase space volume. Given a size for the time step, $\epsilon$, an iteration of the leapfrog method goes as follows:

$$\begin{aligned} \mathbf{p}(\tau + \epsilon/2) &= \mathbf{p}(\tau) - (\epsilon/2)\nabla E(\mathbf{q}(\tau)) \\ \mathbf{q}(\tau + \epsilon) &= \mathbf{q}(\tau) + \epsilon\,\mathbf{p} \\ \mathbf{p}(\tau + \epsilon) &= \mathbf{p}(\tau + \epsilon) - (\epsilon/2)\nabla E(\mathbf{q}(\tau + \epsilon)) \end{aligned} \tag{15}$$

## 2.3  THE STOCHASTIC DYNAMICS METHOD

To create a Markov chain that converges to the distribution of (13), we must interleave leapfrog iterations, which keep $H$ (approximately) constant, with steps that can change $H$. It is convenient for the latter to affect only **p**, since it enters into $H$ in a simple way. This general approach is due to Anderson (1980).

I use stochastic steps of the following form to change $H$:

$$\mathbf{p}' = \alpha\mathbf{p} + (1 - \alpha^2)^{1/2}\mathbf{n} \tag{16}$$

where $0 \leq \alpha < 1$, and **n** is a random vector with components picked independently from Gaussian distributions of mean zero and standard deviation one. One can show that these steps leave the distribution of (13) invariant. Alternating these stochastic steps with dynamical leapfrog steps will therefore sample values for **q** and **p** with close to the desired probabilities. In so far as the discretized dynamics does not keep $H$ exactly constant, however, there will be some degree of bias, which will be eliminated only in the limit as $\epsilon$ goes to zero.

It is best to use a value of $\alpha$ close to one, as this reduces the random walk aspect of the dynamics. If the random term in (16) is omitted, the procedure is equivalent to ordinary batch mode backpropagation learning with momentum.

## 2.4   THE HYBRID MONTE CARLO METHOD

The bias introduced into the stochastic dynamics method by using an approximation to the dynamics is eliminated in the Hybrid Monte Carlo method of Duane, Kennedy, Pendleton, and Roweth (1987).

This method is a variation on the algorithm of Metropolis, *et al* (1953), which generates a Markov chain by considering randomly-selected changes to the state. A change is always accepted if it lowers the energy ($H$), or leaves it unchanged. If it increases the energy, it is accepted with probability $\exp(-\Delta H)$, and is rejected otherwise, with the old state then being repeated.

In the Hybrid Monte Carlo method, candidate changes are produced by picking a random value for p from its distribution given by (13) and then performing some pre-determined number of leapfrog steps. If the leapfrog method were exact, $H$ would be unchanged, and these changes would always be accepted. Since the method is actually only approximate, $H$ sometimes increases, and changes are sometimes rejected, exactly cancelling the bias introduced by the approximation.

Of course, if the errors are very large, the acceptance probability will be very low, and it will take a long time to reach and explore the stationary distribution. To avoid this, we need to choose a step size ($\epsilon$) that is small enough.

## 3   RESULTS ON A TEST PROBLEM

I use the "robot arm" problem of MacKay (1992) for testing. The task is to learn the mapping from two real-valued inputs, $x_1$ and $x_2$, to two real-valued outputs, $y_1$ and $y_2$, given by

$$\bar{y}_1 \;=\; 2.0\cos(x_1) \;+\; 1.3\cos(x_1 + x_2) \tag{17}$$

$$\bar{y}_2 \;=\; 2.0\sin(x_1) \;+\; 1.3\sin(x_1 + x_2) \tag{18}$$

Gaussian noise of mean zero and standard deviation 0.05 is added to $(\bar{y}_1, \bar{y}_2)$ to give the observed position, $(y_1, y_2)$. The training and test sets each consist of 200 cases, with $x_1$ picked randomly from the ranges $[-1.932, -0.453]$ and $[+0.453, +1.932]$, and $x_2$ from the range $[0.534, 3.142]$.

A network with 16 sigmoidal hidden units was used. The output units were linear. Like MacKay, I group weights into three categories — input to hidden, bias to hidden, and hidden/bias to output. MacKay gives separate priors to weights in each category, finding an appropriate value of $\omega$ for each. I fix $\omega$ to one, but multiply each weight by a scale factor associated with its category before using it, giving an equivalent effect. For conventional training with weight decay, I use an analogous scheme with three weight decay constants ($\lambda$ in (6)).

In all cases, I assume that the true value of $\sigma$ is not known. I therefore use (3) for the training set probability, and (5) for the data fit term in conventional training. I set $s_0 = m_0 = 0.1$, which corresponds to a very vague prior for $\sigma$.

## 3.1   PERFORMANCE OF CONVENTIONAL LEARNING

Conventional backpropagation learning was tested on the robot arm problem to gauge how difficult it is to obtain good generalization with standard methods.

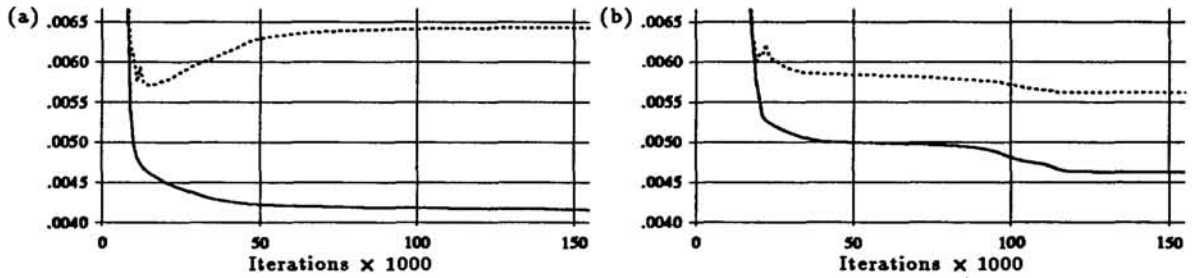

Figure 1: Conventional backpropagation learning — (a) with no weight decay, (b) with carefully-chosen weight decay constants. The solid lines give the squared error on the training data, the dotted lines the squared error on the test data.

Fig. 1(a) shows results obtained without using weight decay. Error on the test set declined initially, but then increased with further training. To achieve good results, the point where the test error reaches its minimum would have to be identified using a separate validation set.

Fig. 1(b) shows results using good weight decay constants, one for each category of weights, taken from the Bayesian runs described below. In this case there is no need to stop learning early, but finding the proper weight decay constants by non-Bayesian methods would be a problem. Again, a validation set seems necessary, as well as considerable computation.

Use of a validation set is wasteful, since data that could otherwise be included in the training set must be excluded. Standard techniques for avoiding this, such as "N-fold" cross-validation, are difficult to apply to neural networks.

## 3.2    PERFORMANCE OF BAYESIAN LEARNING

Bayesian learning was first tested using the unbiased Hybrid Monte Carlo method. The parameter vector in the simulations (q) consisted of the unscaled network weights together with the scale factors for the three weight categories. The actual weight vector (w) was obtained by multiplying each unscaled weight by the scale factor for its category.

Each Hybrid Monte Carlo run consisted of 500 Metropolis steps. For each step, a trajectory consisting of 1000 leapfrog iterations with $\epsilon = 0.00012$ was computed, and accepted or rejected based on the change in $H$ at its end-point. Each run therefore required 500,000 batch gradient evaluations, and took approximately four hours on a machine rated at about 25 MIPS.

Fig. 2(a) shows the training and test error for the early portion of one Hybrid Monte Carlo run. After initially declining, these values fluctuate about an average. Though not apparent in the figure, some quantities (notably the scale factors) require a hundred or more steps to reach their final distribution. The first 250 steps of each run were therefore discarded as not being from the stationary distribution.

Fig. 2(b) shows the training and test set errors produced by networks with weight vectors taken from the last 250 steps of the same run. Also shown is the error on the test set using the *average* of the outputs of all these networks — that is, the estimate given by (10) for the Bayesian prediction of (9). For the run shown, this

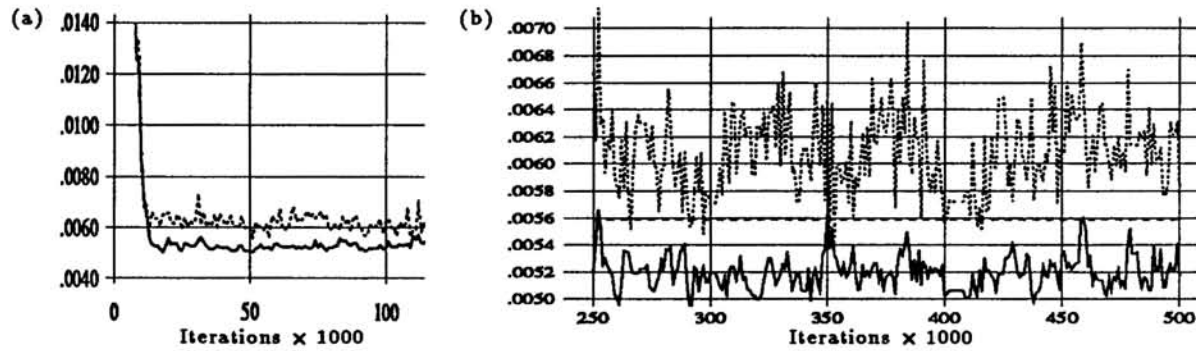

Figure 2: Bayesian learning using Hybrid Monte Carlo — (a) early portion of run, (b) last 250 iterations. The solid lines give the squared error on the training set, the dotted lines the squared error on the test set, for individual networks. The dashed line in (b) is the test error when using the average of the outputs of all 250 networks.

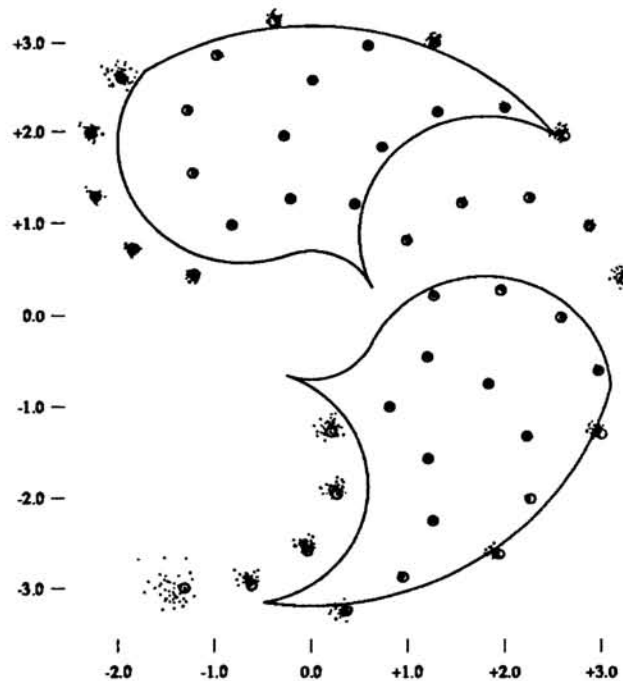

Figure 3: Predictive distribution for outputs. The two regions from which training data was drawn are outlined. Circles indicate the true, noise-free outputs for a grid of cases in the input space. The dots in the vicinity of each circle (often piled on top of it) are the outputs of every fifth network from the last 250 iterations of a Hybrid Monte Carlo run.

test set error using averaged outputs is 0.00559, which is (slightly) better than any results obtained using conventional training. Note that with Bayesian training no validation set is necessary. The analogues of the weight decay constants — the weight scale factors — are found during the course of the simulation.

Another advantage of the Bayesian approach is that it can provide an indication of how uncertain the predictions for test cases are. Fig. 3 demonstrates this. As one would expect, the uncertainty is greater for test cases with inputs outside the region where training data was supplied.

## 3.3   STOCHASTIC DYNAMICS VS. HYBRID MONTE CARLO

The uncorrected stochastic dynamics method will have some degree of systematic bias, due to inexact simulation of the dynamics. Is the amount of bias introduced of any practical importance, however?

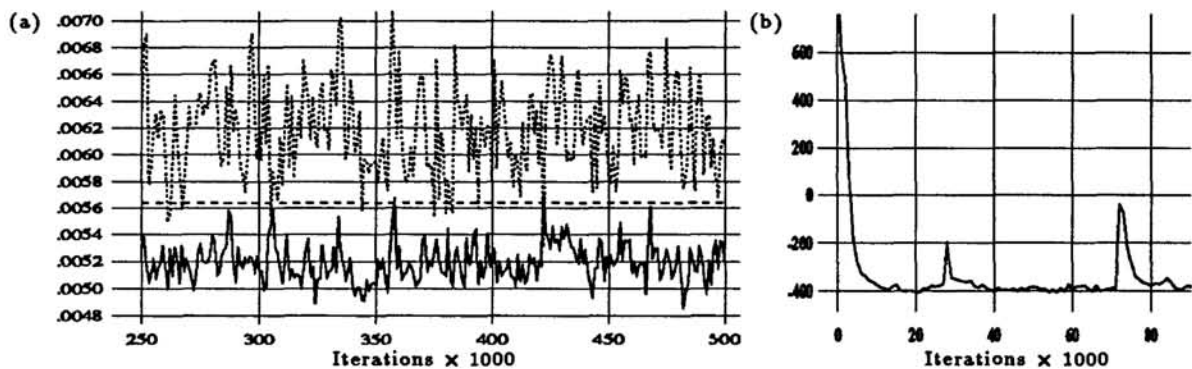

Figure 4: Bayesian learning using uncorrected stochastic dynamics — (a) Training and test error for the last 250 iterations of a run with $\epsilon = 0.00012$, (b) potential energy ($E$) for a run with $\epsilon = 0.00030$. Note the two peaks where the dynamics became unstable.

To help answer this question, the stochastic dynamics method was run with parameters analogous to those used in the Hybrid Monte Carlo runs. The step size of $\epsilon = 0.00012$ used in those runs was chosen to be as large as possible while keeping the number of trajectories rejected low (about 10%). A smaller step size would not give competitive results, so this value was used for the stochastic dynamics runs as well. A value of 0.999 for $\alpha$ in (16) was chosen as being (loosely) equivalent to the use of trajectories 1000 iterations long in the Hybrid Monte Carlo runs.

The results shown in Fig. 4(a) are comparable to those obtained using Hybrid Monte Carlo in Fig. 2(b). Fig. 4(b) shows that with a larger step size the uncorrected stochastic dynamics method becomes unstable. Large step sizes also cause problems for the Hybrid Monte Carlo method, however, as they lead to high rejection rates.

The Hybrid Monte Carlo method may be the more robust choice in some circumstances, but uncorrected stochastic dynamics can also give good results. As it is simpler, the stochastic dynamics method may be better for hardware implementation, and is a more plausible starting point for any attempt to relate Bayesian methods to biology. Numerous other variations on these methods are possible as well, some of which are discussed in (Neal, 1992).

## References

Andersen, H. C. (1980) "Molecular dynamics simulations at constant pressure and/or temperature", *Journal of Chemical Physics*, vol. 72, pp. 2384-2393.

Buntine, W. L. and Weigend, A. S. (1991) "Bayesian back-propagation", *Complex Systems*, vol. 5, pp. 603-643.

Duane, S., Kennedy, A. D., Pendleton, B. J., and Roweth, D. (1987) "Hybrid Monte Carlo", *Physics Letters B*, vol. 195, pp. 216-222.

MacKay, D. J. C. (1992) "A practical Bayesian framework for backpropagation networks", *Neural Computation*, vol. 4, pp. 448-472.

Metropolis, N., Rosenbluth, A. W., Rosenbluth, M. N., Teller, A. H., and Teller, E. (1953) "Equation of state calculations by fast computing machines", *Journal of Chemical Physics*, vol. 21, pp. 1087-1092.

Neal, R. M. (1992) "Bayesian training of backpropagation networks by the hybrid Monte Carlo method", CRG-TR-92-1, Dept. of Computer Science, University of Toronto.
